# Describing Visual Scenes using Transformed Dirichlet Processes

**Erik B. Sudderth, Antonio Torralba, William T. Freeman, and Alan S. Willsky**
Department of Electrical Engineering and Computer Science
Massachusetts Institute of Technology
*esuddert@mit.edu, torralba@csail.mit.edu, billf@mit.edu, willsky@mit.edu*

## Abstract

Motivated by the problem of learning to detect and recognize objects with minimal supervision, we develop a hierarchical probabilistic model for the spatial structure of visual scenes. In contrast with most existing models, our approach explicitly captures uncertainty in the *number* of object instances depicted in a given image. Our scene model is based on the transformed Dirichlet process (TDP), a novel extension of the hierarchical DP in which a set of stochastically transformed mixture components are shared between multiple groups of data. For visual scenes, mixture components describe the spatial structure of visual features in an object–centered coordinate frame, while transformations model the object positions in a particular image. Learning and inference in the TDP, which has many potential applications beyond computer vision, is based on an empirically effective Gibbs sampler. Applied to a dataset of partially labeled street scenes, we show that the TDP's inclusion of spatial structure improves detection performance, flexibly exploiting partially labeled training images.

## 1 Introduction

In this paper, we develop methods for analyzing the features composing a *visual scene*, thereby localizing and categorizing the objects in an image. We would like to design learning algorithms that exploit relationships among multiple, partially labeled object categories during training. Working towards this goal, we propose a hierarchical probabilistic model for the expected spatial locations of objects, and the appearance of visual features corresponding to each object. Given a new image, our model provides a globally coherent explanation for the observed scene, including estimates of the location and category of an *a priori* unknown number of objects.

This generative approach is motivated by the pragmatic need for learning algorithms which require little manual supervision and labeling. While discriminative models may produce accurate classifiers, they typically require very large training sets even for relatively simple categories [1]. In contrast, generative approaches can discover large, visually salient categories (such as foliage and buildings [2]) without supervision. Partial segmentations can then be used to learn semantically interesting categories (such as cars and pedestrians) which are less visually distinctive, or present in fewer training images. Moreover, generative models provide a natural framework for learning contextual relationships between objects, and transferring knowledge between related, but distinct, visual scenes.

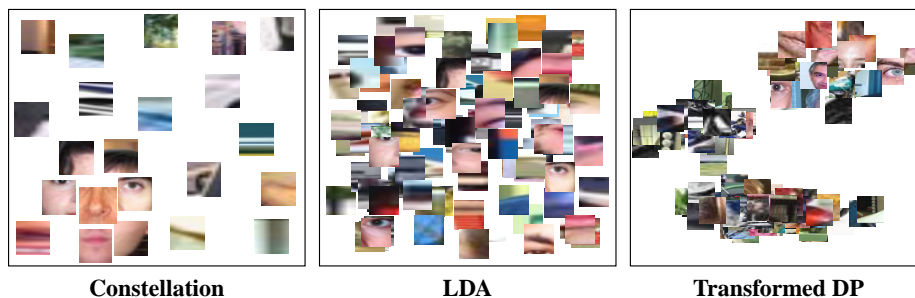

| Constellation | LDA | Transformed DP |

Figure 1: A scene with faces as described by three generative models. *Constellation:* Fixed parts of a single face in unlocalized clutter. *LDA:* Bag of unlocalized face and background features. *TDP:* Spatially localized clusters of background clutter, and one or more faces (in this case, the sample contains one face and two background clusters). *Note:* The LDA and TDP images are sampled from models learned from training images, while the Constellation image is a hand-constructed illustration.

The principal challenge in developing hierarchical models for scenes is specifying tractable, scalable methods for handling uncertainty in the number of objects. This issue is entirely ignored by most existing models. We address this problem using Dirichlet processes [3], a tool from nonparametric Bayesian analysis for learning mixture models whose number of components is not fixed, but instead estimated from data. In particular, we extend the recently proposed *hierarchical Dirichlet process* (HDP) [4, 5] framework to allow more flexible sharing of mixture components between images. The resulting *transformed Dirichlet process* (TDP) is naturally suited to our scene understanding application, as well as many other domains where "style and content" are combined to produce the observed data [6].

We begin in Sec. 2 by reviewing several related generative models for objects and scenes. Sec. 3 then introduces Dirichlet processes and develops the TDP model, including MCMC methods for learning and inference. We specialize the TDP to visual scenes in Sec. 4, and conclude in Sec. 5 by demonstrating object recognition and segmentation in street scenes.

## 2 Generative Models for Objects and Scenes

*Constellation models* [7] describe single objects via the appearance of a fixed, and typically small, set of spatially constrained parts (see Fig. 1). Although they can successfully recognize objects in cluttered backgrounds, they do not directly provide a mechanism for detecting multiple object instances. In addition, it seems difficult to generalize the fixed set of constellation parts to problems where the number of objects is uncertain.

*Grammars*, and related rule–based systems, were one of the earliest approaches to scene understanding [8]. More recently, distributions over hierarchical tree–structured partitions of image pixels have been used to segment simple scenes [9, 10]. In addition, an *image parsing* [11] framework has been proposed which explains an image using a set of regions generated by generic or object–specific processes. While this model allows uncertainty in the number of regions, and hence the number of objects, the high dimensionality of the model state space requires good, discriminatively trained bottom–up proposal distributions for acceptable MCMC performance. We also note that the BLOG language [12] provides a promising framework for reasoning about unknown objects. As of yet, however, the computational tools needed to apply BLOG to large–scale applications are unavailable.

Inspired by techniques from the text analysis literature, several recent papers analyze scenes using a spatially unstructured *bag of features* extracted from local image patches (see Fig. 1). In particular, *latent Dirichlet allocation* (LDA) [13] describes the features $x_{ji}$ in image $j$ using a $K$ component mixture model with parameters $\theta_k$. Each image reuses these same mixture parameters in different proportions $\pi_j$ (see the graphical model of Fig. 2). By appropriately defining these shared mixtures, LDA may be used to discover object cat-

egories from images of single objects [2], categorize natural scenes [14], and (with a slight extension) parse presegmented captioned images [15].

While these LDA models are sometimes effective, their neglect of spatial structure ignores valuable information which is critical in challenging object detection tasks. We recently proposed a hierarchical extension of LDA which learns shared parts describing the internal structure of objects, and contextual relationships among known groups of objects [16]. The *transformed Dirichlet process* (TDP) addresses a key limitation of this model by allowing uncertainty in the number and identity of the objects depicted in each image. As detailed in Sec. 4 and illustrated in Fig. 1, the TDP effectively provides a *textural* model in which locally unstructured clumps of features are given global spatial structure by the inferred set of objects underlying each scene.

## 3 Hierarchical Modeling using Dirichlet Processes

In this section, we review Dirichlet process mixture models (Sec. 3.1) and previously proposed hierarchical extensions (Sec. 3.2). We then introduce the *transformed Dirichlet process* (TDP) (Sec. 3.3), and discuss Monte Carlo methods for learning TDPs (Sec. 3.4).

### 3.1 Dirichlet Process Mixture Models

Let $\theta$ denote a parameter taking values in some space $\Theta$, and $H$ be a measure on $\Theta$. A *Dirichlet process* (DP), denoted by $\mathrm{DP}(\gamma, H)$, is then a distribution over measures on $\Theta$, where the concentration parameter $\gamma$ controls the similarity of samples $G \sim \mathrm{DP}(\gamma, H)$ to the base measure $H$. Samples from DPs are discrete with probability one, a property highlighted by the following *stick–breaking construction* [4]:

$$G(\theta) = \sum_{k=1}^{\infty} \beta_k \delta(\theta, \theta_k) \qquad \beta_k' \sim \mathrm{Beta}(1, \gamma) \qquad \beta_k = \beta_k' \prod_{\ell=1}^{k-1} (1 - \beta_\ell') \quad (1)$$

Each parameter $\theta_k \sim H$ is independently sampled, while the weights $\boldsymbol{\beta} = (\beta_1, \beta_2, \ldots)$ use Beta random variables to partition a unit–length "stick" of probability mass.

In nonparametric Bayesian statistics, DPs are commonly used as prior distributions for mixture models with an unknown number of components [3]. Let $F(\theta)$ denote a family of distributions parameterized by $\theta$. Given $G \sim \mathrm{DP}(\gamma, H)$, each observation $x_i$ from an exchangeable data set $\mathbf{x}$ is generated by first choosing a parameter $\bar{\theta}_i \sim G$, and then sampling $x_i \sim F(\bar{\theta}_i)$. Computationally, this process is conveniently described by a set $\mathbf{z}$ of independently sampled variables $z_i \sim \mathrm{Mult}(\boldsymbol{\beta})$ indicating the component of the mixture $G(\theta)$ (see eq. (1)) associated with each data point $x_i \sim F(\theta_{z_i})$.

Integrating over $G$, the indicator variables $\mathbf{z}$ demonstrate an important clustering property. Letting $n_k$ denote the number of times component $\theta_k$ is chosen by the first $(i-1)$ samples,

$$p\left(z_i \mid z_1, \ldots, z_{i-1}, \gamma\right) = \frac{1}{\gamma + i - 1} \left[ \sum_k n_k \delta(z_i, k) + \gamma \delta(z_i, \bar{k}) \right] \quad (2)$$

Here, $\bar{k}$ indicates a previously unused mixture component (*a priori*, all unused components are equivalent). This process is sometimes described by analogy to a Chinese restaurant in which the (infinite collection of) tables correspond to the mixture components $\theta_k$, and customers to observations $x_i$ [4]. Customers are social, tending to sit at tables with many other customers (observations), and each table shares a single dish (parameter).

### 3.2 Hierarchical Dirichlet Processes

In many domains, there are several groups of data produced by related, but distinct, generative processes. For example, in this paper's applications each group is an image, and the data are visual features composing a scene. Given $J$ groups of data, let $\mathbf{x}_j = (x_{j1}, \ldots, x_{jn_j})$ denote the $n_j$ exchangeable data points in group $j$.

*Hierarchical Dirichlet processes* (HDPs) [4, 5] describe grouped data with a coupled set of

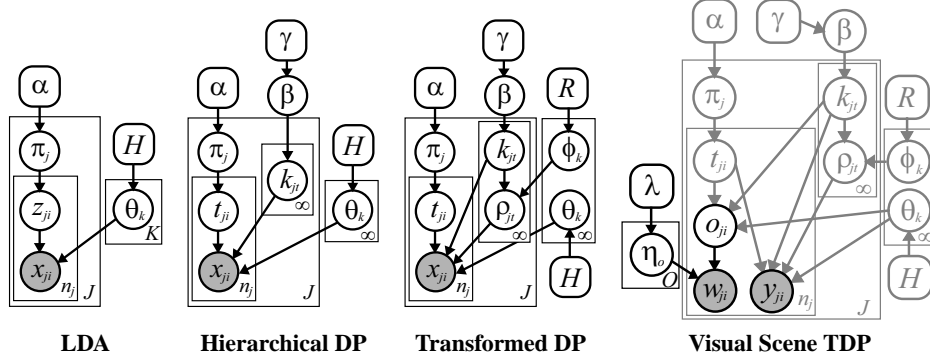

**LDA**　　　　**Hierarchical DP**　　　　**Transformed DP**　　　　**Visual Scene TDP**

Figure 2: Graphical representations of the LDA, HDP, and TDP models for sharing mixture components $\theta_k$, with proportions $\pi_j$, among $J$ groups of exchangeable data $\mathbf{x}_j = (x_{j1}, \ldots, x_{jn_j})$. LDA directly assigns observations $x_{ji}$ to clusters via indicators $z_{ji}$. HDP and TDP models use "table" indicators $t_{ji}$ as an intermediary between observations and assignments $k_{jt}$ to an infinite global mixture with weights $\beta$. TDPs augment each table $t$ with a transformation $\rho_{jt}$ sampled from a distribution parameterized by $\phi_{k_{jt}}$. Specializing the TDP to visual scenes (right), we model the position $y_{ji}$ and appearance $w_{ji}$ of features using distributions $\eta_o$ indexed by unobserved object categories $o_{ji}$.

mixture models. To construct an HDP, a global probability measure $G_0 \sim \mathrm{DP}(\gamma, H)$ is first chosen to define a set of shared mixture components. A measure $G_j \sim \mathrm{DP}(\alpha, G_0)$ is then independently sampled for each group. Because $G_0$ is discrete (as in eq. (1)), groups $G_j$ will reuse the same mixture components $\theta_k$ in different proportions:

$$G_j(\theta) = \sum_{k=1}^{\infty} \pi_{jk}\delta(\theta, \theta_k) \qquad\qquad \boldsymbol{\pi}_j \sim \mathrm{DP}(\alpha, \boldsymbol{\beta}) \qquad\qquad (3)$$

In this construction, shared components improve generalization when learning from few examples, while distinct mixture weights capture differences between groups.

The generative process underlying HDPs may be understood in terms of an extension of the DP analogy known as the *Chinese restaurant franchise* [4]. Each group defines a separate restaurant in which customers (observations) $x_{ji}$ sit at tables $t_{ji}$. Each table shares a single dish (parameter) $\theta$, which is ordered from a menu $G_0$ shared among restaurants (groups). Letting $k_{jt}$ indicate the parameter $\theta_{k_{jt}}$ assigned to table $t$ in group $j$, we may integrate over $G_0$ and $G_j$ (as in eq. (2)) to find the conditional distributions of these indicator variables:

$$p\left(t_{ji} \mid t_{j1}, \ldots, t_{ji-1}, \alpha\right) \propto \sum_t n_{jt}\delta(t_{ji}, t) + \alpha\delta(t_{ji}, \bar{t}) \qquad\qquad (4)$$

$$p\left(k_{jt} \mid \mathbf{k}_1, \ldots, \mathbf{k}_{j-1}, k_{j1}, \ldots, k_{jt-1}, \gamma\right) \propto \sum_k m_k\delta(k_{jt}, k) + \gamma\delta(k_{jt}, \bar{k}) \qquad\qquad (5)$$

Here, $m_k$ is the number of tables previously assigned to $\theta_k$. As before, customers prefer tables $t$ at which many customers $n_{jt}$ are already seated (eq. (4)), but sometimes choose a new table $\bar{t}$. Each new table is assigned a dish $k_{j\bar{t}}$ according to eq. (5). Popular dishes are more likely to be ordered, but a new dish $\theta_{\bar{k}} \sim H$ may also be selected.

The HDP generative process is summarized in the graphical model of Fig. 2. Given the assignments $\mathbf{t}_j$ and $\mathbf{k}_j$ for group $j$, observations are sampled as $x_{ji} \sim F(\theta_{z_{ji}})$, where $z_{ji} = k_{jt_{ji}}$ indexes the shared parameters assigned to the table associated with $x_{ji}$.

### 3.3 Transformed Dirichlet Processes

In the HDP model of Fig. 2, the group distributions $G_j$ are derived from the global distribution $G_0$ by resampling the mixture weights from a Dirichlet process (see eq. (3)), leaving the component parameters $\theta_k$ unchanged. In many applications, however, it is difficult to define $\theta$ so that parameters may be exactly reused between groups. Consider, for example,

a Gaussian distribution describing the location at which object features are detected in an image. While the covariance of that distribution may stay relatively constant across object instances, the mean will change dramatically from image to image (group to group), depending on the objects' position relative to the camera.

Motivated by these difficulties, we propose the *Transformed Dirichlet Process* (TDP), an extension of the HDP in which global mixture components undergo a set of random transformations before being reused in each group. Let $\rho$ denote a transformation of the parameter vector $\theta \in \Theta$, $\phi \in \Phi$ the parameters of a distribution $Q$ over transformations, and $R$ a measure on $\Phi$. We begin by augmenting the DP stick–breaking construction of eq. (1) to create a global measure describing both parameters and transformations:

$$G_0(\theta, \rho) = \sum_{k=1}^{\infty} \beta_k \delta(\theta, \theta_k) q(\rho \mid \phi_k) \qquad \theta_k \sim H \qquad \phi_k \sim R \qquad (6)$$

As before, $\boldsymbol{\beta}$ is sampled from a stick–breaking process with parameter $\gamma$. For each group, we then sample a measure $G_j \sim \mathrm{DP}(\alpha, G_0)$. Marginalizing over transformations $\rho$, $G_j(\theta)$ reuses parameters from $G_0(\theta)$ exactly as in eq. (3). Because samples from DPs are discrete, the joint measure for group $j$ then has the following form:

$$G_j(\theta, \rho) = \sum_{k=1}^{\infty} \pi_{jk} \delta(\theta, \theta_k) \left[ \sum_{\ell=1}^{\infty} \omega_{jk\ell} \delta(\rho, \rho_{jk\ell}) \right] \qquad \sum_{\ell=1}^{\infty} \omega_{jk\ell} = 1 \qquad (7)$$

Note that within the $j^{th}$ group, each shared parameter vector $\theta_k$ may potentially be reused multiple times with different transformations $\rho_{jk\ell}$. Conditioning on $\theta_k$, it can be shown that $G_j(\rho \mid \theta_k) \sim \mathrm{DP}(\alpha\beta_k, Q(\phi_k))$, so that the proportions $\boldsymbol{\omega}_{jk}$ of features associated with each transformation of $\theta_k$ follow a stick–breaking process with parameter $\alpha\beta_k$.

Each observation $x_{ji}$ is now generated by sampling $(\bar{\theta}_{ji}, \bar{\rho}_{ji}) \sim G_j$, and then choosing $x_{ji} \sim F(\bar{\theta}_{ji}, \bar{\rho}_{ji})$ from a distribution which transforms $\bar{\theta}_{ji}$ by $\bar{\rho}_{ji}$. Although the global family of transformation distributions $Q(\phi)$ is typically non–atomic, the discreteness of $G_j$ ensures that transformations are shared between observations within group $j$.

Computationally, the TDP is more conveniently described via an extension of the Chinese restaurant franchise analogy (see Fig. 2). As before, customers (observations) $x_{ji}$ sit at tables $t_{ji}$ according to the clustering bias of eq. (4), and new tables choose dishes according to their popularity across the franchise (eq. (5)). Now, however, the dish (parameter) $\theta_{k_{jt}}$ at table $t$ is seasoned (transformed) according to $\rho_{jt} \sim q(\rho_{jt} \mid \phi_{k_{jt}})$. Each time a dish is ordered, the recipe is seasoned differently.

### 3.4   Learning via Gibbs Sampling

To learn the parameters of a TDP, we extend the HDP Gibbs sampler detailed in [4]. The simplest implementation samples table assignments $\mathbf{t}$, cluster assignments $\mathbf{k}$, transformations $\boldsymbol{\rho}$, and parameters $\boldsymbol{\theta}, \boldsymbol{\phi}$. Let $\mathbf{t}^{-ji}$ denote all table assignments excluding $t_{ji}$, and define $\mathbf{k}^{-jt}, \boldsymbol{\rho}^{-jt}$ similarly. Using the Markov properties of the TDP (see Fig. 2), we have

$$p\left(t_{ji} = t \mid \mathbf{t}^{-ji}, \mathbf{k}, \boldsymbol{\rho}, \boldsymbol{\theta}, \mathbf{x}\right) \propto p\left(t \mid \mathbf{t}^{-ji}\right) f\left(x_{ji} \mid \theta_{k_{jt}}, \rho_{jt}\right) \qquad (8)$$

The first term is given by eq. (4). For a fixed set of transformations $\boldsymbol{\rho}$, the second term is a simple likelihood evaluation for existing tables, while new tables may be evaluated by marginalizing over possible cluster assignments (eq. (5)).

Because cluster assignments $k_{jt}$ and transformations $\rho_{jt}$ are strongly coupled in the posterior, a blocked Gibbs sampler which jointly resamples them converges much more rapidly:

$$p\left(k_{jt} = k, \rho_{jt} \mid \mathbf{k}^{-jt}, \boldsymbol{\rho}^{-jt}, \mathbf{t}, \boldsymbol{\theta}, \boldsymbol{\phi}, \mathbf{x}\right) \propto p\left(k \mid \mathbf{k}^{-jt}\right) q\left(\rho_{jt} \mid \phi_k\right) \prod_{t_{ji}=t} f\left(x_{ji} \mid \theta_k, \rho_{jt}\right)$$

For the models considered in this paper, $F$ is conjugate to $Q$ for any fixed observation value. We may thus analytically integrate over $\rho_{jt}$ and, combined with eq. (5), sample a

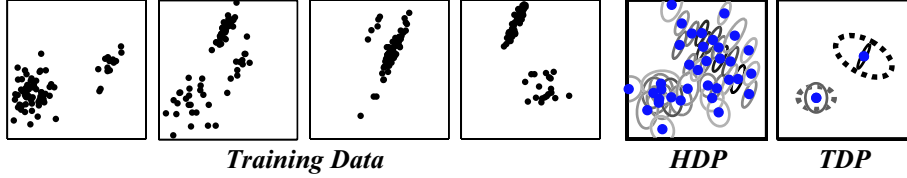

*Training Data*      *HDP*     *TDP*

Figure 3: Comparison of hierarchical models learned via Gibbs sampling from synthetic 2D data. *Left:* Four of 50 "images" used for training. *Center:* Global distribution $G_0(\theta)$ for the HDP, where ellipses are covariance estimates and intensity is proportional to prior probability. *Right:* Global TDP distribution $G_0(\theta, \rho)$ over both clusters $\theta$ (solid) and translations $\rho$ of those clusters (dashed).

new cluster assignment $\bar{k}_{jt}$. Conditioned on $\bar{k}_{jt}$, we again use conjugacy to sample $\bar{\rho}_{jt}$. We also choose the parameter priors $H$ and $R$ to be conjugate to $Q$ and $F$, respectively, so that standard formulas may be used to resample $\boldsymbol{\theta}, \boldsymbol{\phi}$.

## 4 Transformed Dirichlet Processes for Visual Scenes

### 4.1 Context–Free Modeling of Multiple Object Categories

In this section, we adapt the TDP model of Sec. 3.3 to describe the spatial structure of visual scenes. Groups $j$ now correspond to training, or test, images. For the moment, we assume that the observed data $x_{ji} = (o_{ji}, y_{ji})$, where $y_{ji}$ is the position of a feature corresponding to object category $o_{ji}$, and the number of object categories $O$ is known (see Fig. 2). We then choose cluster parameters $\theta_k = (\bar{o}_k, \mu_k, \Lambda_k)$ to describe the mean $\mu_k$ and covariance $\Lambda_k$ of a Gaussian distribution over feature positions, as well as the *single* object category $\bar{o}_k$ assigned to *all* observations sampled from that cluster. Although this cluster parameterization does not capture contextual relationships between object categories, the results of Sec. 5 demonstrate that it nevertheless provides an effective model of the spatial variability of individual categories across many different scenes.

To model the variability in object location from image to image, transformation parameters $\rho_{jt}$ are defined to *translate* feature position relative to that cluster's "canonical" mean $\mu_k$:

$$p\left(o_{ji}, y_{ji} \mid t_{ji} = t, \mathbf{k}_j, \boldsymbol{\rho}_j, \boldsymbol{\theta}\right) = \delta(o_{ji}, \bar{o}_{k_{jt}}) \times \mathcal{N}\left(y_{ji}; \mu_{k_{jt}} + \rho_{jt}, \Lambda_{k_{jt}}\right) \qquad (9)$$

We note that there is a different translation $\rho_{jt}$ associated with each table $t$, allowing the same object cluster to be reused at multiple locations within a single image. This flexibility, which is not possible with HDPs, is critical to accurately modeling visual scenes. Density models for spatial transformations have been previously used to recognize isolated objects [17], and estimate layered decompositions of video sequences [18]. In contrast, the proposed TDP models the variability of object positions across scenes, and couples this with a nonparametric prior allowing uncertainty in the number of objects.

To ensure that the TDP scene model is identifiable, we define $p\left(\rho_{jt} \mid \mathbf{k}_j, \boldsymbol{\phi}\right)$ to be a zero–mean Gaussian with covariance $\phi_{k_{jt}}$. The parameter prior $R$ is uniform across object categories, while $R$ and $H$ both use inverse–Wishart position distributions, weakly biased towards moderate covariances. Fig. 3 shows a 2D synthetic example based on a single object category ($O = 1$). Following 100 Gibbs sampling iterations, the TDP correctly discovers that the data is composed of elongated "bars" in the upper right, and round "blobs" in the lower left. In contrast, the learned HDP uses a large set of global clusters to discretize the transformations underlying the data, and thus generalizes poorly to new translations.

### 4.2 Detecting Objects from Image Features

To apply the TDP model of Sec. 4.1 to images, we must learn the relationship between object categories and visual features. As in [2, 16], we obtain discrete features by vector quantizing SIFT descriptors [19] computed over locally adapted elliptical regions. To improve discriminative power, we divide these elliptical regions into three groups (roughly circu-

lar, and horizontally or vertically elongated) prior to quantizing SIFT values, producing a discrete vocabulary with $1800$ appearance "words". Given the density of feature detection, these descriptors essentially provide a multiscale over–segmentation of the image.

We assume that the appearance $w_{ji}$ of each detected feature is independently sampled conditioned on the underlying object category $o_{ji}$ (see Fig. 2). Placing a symmetric Dirichlet prior, with parameter $\lambda$, on each category's multinomial appearance distribution $\eta_o$,

$$p\left(w_{ji} = b \mid o_{ji} = o, \mathbf{w}^{-ji}, \mathbf{t}, \mathbf{k}, \boldsymbol{\theta}\right) \propto c_{bo} + \lambda \qquad (10)$$

where $c_{bo}$ is the number of times feature $b$ is currently assigned to object $o$. Because a single object category is associated with each cluster, the Gibbs sampler of Sec. 3.4 may be easily adapted to this case by incorporating eq. (10) into the assignment likelihoods.

## 5 Analyzing Street Scenes

To demonstrate the potential of our TDP scene model, we consider a set of street scene images (250 training, 75 test) from the MIT-CSAIL database. These images contain three "objects": buildings, cars (side views), and roads. All categories were labeled in 112 images, while in the remainder only cars were segmented. Training from semi–supervised data is accomplished by restricting object category assignments for segmented features.

Fig. 4 shows the four global object clusters learned following 100 Gibbs sampling iterations. There is one elongated car cluster, one large building cluster, and two road clusters with differing shapes. Interestingly, the model has automatically determined that building features occur in large homogeneous patches, while road features are sparse and better described by many smaller transformed clusters. To segment test images, we run the Gibbs sampler for 50 iterations from each of 10 random initializations. Fig. 4 shows segmentations produced by averaging these samples, as well as transformed clusters from the final iteration. Qualitatively, results are typically good, although foliage is often mislabeled as road due to the textural similarities with features detected in shadows across roads.

For comparison, we also trained an LDA model based solely on feature appearance, allowing three topics per object category and again using object labels to restrict the Gibbs sampler's assignments [16]. As shown by the ROC curves of Fig. 4, our TDP model of spatial scene structure significantly improves segmentation performance. In addition, through the set of transformed car clusters generated by the Gibbs sampler, the TDP explicitly estimates the number of object *instances* underlying each image. These detections, which are not possible using LDA, are based on a single global parsing of the scene which automatically estimates object locations without a "sliding window" [1].

## 6 Discussion

We have developed the transformed Dirichlet process, a hierarchical model which shares a set of stochastically transformed clusters among groups of data. Applied to visual scenes, TDPs provide a model of spatial structure which allows the number of objects generating an image to be automatically inferred, and lead to improved detection performance. We are currently investigating extensions of the basic TDP scene model presented in this paper which describe the internal structure of objects, and also incorporate richer contextual cues.

**Acknowledgments**

Funding provided by the National Geospatial-Intelligence Agency NEGI-1582-04-0004, the National Science Foundation NSF-IIS-0413232, the ARDA VACE program, and a grant from BAE Systems.

## References

[1] P. Viola and M. J. Jones. Robust real–time face detection. *IJCV*, 57(2):137–154, 2004.

[2] J. Sivic, B. C. Russell, A. A. Efros, A. Zisserman, and W. T. Freeman. Discovering objects and their location in images. In *ICCV*, 2005.

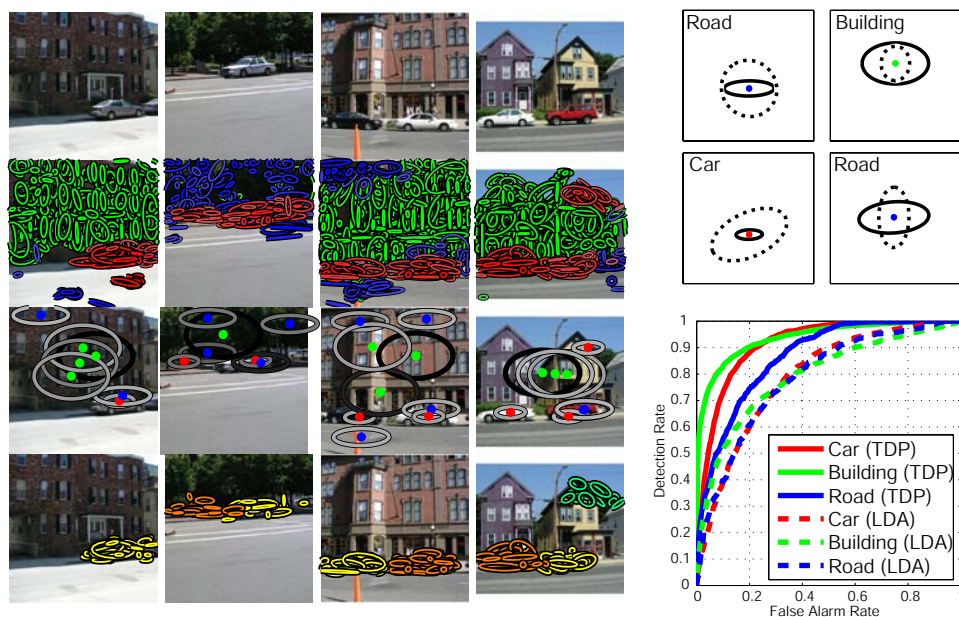

Figure 4: TDP analysis of street scenes containing cars (red), buildings (green), and roads (blue). *Top right:* Global model $G_0$ describing object shape (solid) and expected transformations (dashed). *Bottom right:* ROC curves comparing TDP feature segmentation performance to an LDA model of feature appearance. *Left:* Four test images (first row), estimated segmentations of features into object categories (second row), transformed global clusters associated with each image interpretation (third row), and features assigned to different instances of the transformed car cluster (fourth row).

[3] M. D. Escobar and M. West. Bayesian density estimation and inference using mixtures. *J. Amer. Stat. Assoc.*, 90(430):577–588, June 1995.

[4] Y. W. Teh, M. I. Jordan, M. J. Beal, and D. M. Blei. Hierarchical Dirichlet processes. Technical Report 653, U.C. Berkeley Statistics, October 2004.

[5] Y. W. Teh, M. I. Jordan, M. J. Beal, and D. M. Blei. Hierarchical Dirichlet processes. In *NIPS 17*, pages 1385–1392. MIT Press, 2005.

[6] J. B. Tenenbaum and W. T. Freeman. Separating style and content with bilinear models. *Neural Comp.*, 12:1247–1283, 2000.

[7] L. Fei-Fei, R. Fergus, and P. Perona. A Bayesian approach to unsupervised one-shot learning of object categories. In *ICCV*, volume 2, pages 1134–1141, 2003.

[8] J. M. Tenenbaum and H. G. Barrow. Experiments in interpretation-guided segmentation. *Artif. Intel.*, 8:241–274, 1977.

[9] A. J. Storkey and C. K. I. Williams. Image modeling with position-encoding dynamic trees. *IEEE Trans. PAMI*, 25(7):859–871, July 2003.

[10] J. M. Siskind et al. Spatial random tree grammars for modeling hierarchal structure in images. Submitted to IEEE Tran. PAMI, 2004.

[11] Z. Tu, X. Chen, A. L. Yuille, and S. C. Zhu. Image parsing: Unifying segmentation, detection, and recognition. In *ICCV*, volume 1, pages 18–25, 2003.

[12] B. Milch, B. Marthi, S. Russell, D. Sontag, D. L. Ong, and A. Kolobov. BLOG: Probabilistic models with unknown objects. In *IJCAI 19*, pages 1352–1359, 2005.

[13] D. M. Blei, A. Y. Ng, and M. I. Jordan. Latent Dirichlet allocation. *JMLR*, 3:993–1022, 2003.

[14] L. Fei-Fei and P. Perona. A Bayesian hierarchical model for learning natural scene categories. In *CVPR*, volume 2, pages 524–531, 2005.

[15] K. Barnard et al. Matching words and pictures. *JMLR*, 3:1107–1135, 2003.

[16] E. B. Sudderth, A. Torralba, W. T. Freeman, and A. S. Willsky. Learning hierarchical models of scenes, objects, and parts. In *ICCV*, 2005.

[17] E. G. Miller, N. E. Matsakis, and P. A. Viola. Learning from one example through shared densities on transforms. In *CVPR*, volume 1, pages 464–471, 2000.

[18] N. Jojic and B. J. Frey. Learning flexible sprites in video layers. In *CVPR*, volume 1, pages 199–206, 2001.

[19] D. G. Lowe. Distinctive image features from scale–invariant keypoints. *IJCV*, 60(2):91–110, 2004.
